# From PAC-Bayes Bounds to $\mathrm{KL}$ Regularization

**Pascal Germain, Alexandre Lacasse, François Laviolette, Mario Marchand, Sara Shanian**
Department of Computer Science and Software Engineering
Laval University, Québec (QC), Canada
`firstname.secondname@ift.ulaval.ca`

## Abstract

We show that convex KL-regularized objective functions are obtained from a PAC-Bayes risk bound when using convex loss functions for the stochastic Gibbs classifier that upper-bound the standard zero-one loss used for the weighted majority vote. By restricting ourselves to a class of posteriors, that we call *quasi uniform*, we propose a simple coordinate descent learning algorithm to minimize the proposed KL-regularized cost function. We show that standard $\ell_p$-regularized objective functions currently used, such as ridge regression and $\ell_p$-regularized boosting, are obtained from a relaxation of the KL divergence between the quasi uniform posterior and the uniform prior. We present numerical experiments where the proposed learning algorithm generally outperforms ridge regression and AdaBoost.

## 1 Introduction

What should a learning algorithm optimize on the training data in order to give classifiers having the smallest possible true risk? Many different specifications of what should be optimized on the training data have been provided by using different inductive principles. But the universally accepted guarantee on the true risk, however, always comes with a so-called risk bound that holds uniformly over a set of classifiers. Since a risk bound can be computed from what a classifier achieves on the training data, it automatically suggests that learning algorithms should find a classifier that minimizes a tight risk (upper) bound.

Among the data-dependent bounds that have been proposed recently, the PAC-Bayes bounds [6, 8, 4, 1, 3] seem to be especially tight. These bounds thus appear to be a good starting point for the design of a bound-minimizing learning algorithm. In that respect, [4, 5, 3] have proposed to use isotropic Gaussian posteriors over the space of linear classifiers. But a computational drawback of this approach is the fact the Gibbs empirical risk is not a quasi-convex function of the parameters of the posterior. Consequently, the resultant PAC-Bayes bound may have several local minima for certain data sets—thus giving an intractable optimization problem in the general case. To avoid such computational problems, we propose here to use convex loss functions for stochastic Gibbs classifiers that upper-bound the standard zero-one loss used for the weighted majority vote. By restricting ourselves to a class of posteriors, that we call *quasi uniform*, we propose a simple coordinate descent learning algorithm to minimize the proposed KL-regularized cost function. We show that there are no loss of discriminative power by restricting the posterior to be quasi uniform. We also show that standard $\ell_p$-regularized objective functions currently used, such as ridge regression and $\ell_p$-regularized boosting, are obtained from a relaxation of the KL divergence between the quasi uniform posterior and the uniform prior. We present numerical experiments where the proposed learning algorithm generally outperforms ridge regression and AdaBoost [7].

## 2 Basic Definitions

We consider binary classification problems where the input space $\mathcal{X}$ consists of an arbitrary subset of $\mathbb{R}^d$ and the output space $\mathcal{Y} = \{-1, +1\}$. An *example* is an input-output $(\mathbf{x}, y)$ pair where $\mathbf{x} \in \mathcal{X}$ and $y \in \mathcal{Y}$. Throughout the paper, we adopt the PAC setting where each example $(\mathbf{x}, y)$ is drawn according to a fixed, but unknown, distribution $D$ on $\mathcal{X} \times \mathcal{Y}$.

The *risk* $R(h)$ of any classifier $h \colon \mathcal{X} \to \mathcal{Y}$ is defined as the probability that $h$ misclassifies an example drawn according to $D$. Given a training set $S$ of $m$ examples, the *empirical risk* $R_S(h)$ of any classifier $h$ is defined by the frequency of training errors of $h$ on $S$. Hence

$$R(h) \stackrel{\text{def}}{=} \operatorname*{\mathbf{E}}_{(\mathbf{x},y) \sim D} I(h(\mathbf{x}) \neq y) \quad ; \quad R_S(h) \stackrel{\text{def}}{=} \frac{1}{m} \sum_{i=1}^{m} I(h(\mathbf{x}_i) \neq y_i) \,,$$

where $I(a) = 1$ if predicate $a$ is true and $0$ otherwise.

After observing the training set $S$, the task of the learner is to choose a *posterior* distribution $Q$ over a space $\mathcal{H}$ of classifiers such that the $Q$-weighted majority vote classifier $B_Q$ will have the smallest possible risk. On any input example $\mathbf{x}$, the output $B_Q(\mathbf{x})$ of the majority vote classifier $B_Q$ (sometimes called the Bayes classifier) is given by

$$B_Q(\mathbf{x}) \stackrel{\text{def}}{=} \operatorname{sgn} \left[ \operatorname*{\mathbf{E}}_{h \sim Q} h(\mathbf{x}) \right],$$

where $\operatorname{sgn}(s) = +1$ if $s > 0$ and $\operatorname{sgn}(s) = -1$ otherwise. The output of the deterministic majority vote classifier $B_Q$ is closely related to the output of a stochastic classifier called the *Gibbs* classifier $G_Q$. To classify an input example $\mathbf{x}$, the Gibbs classifier $G_Q$ chooses randomly a (deterministic) classifier $h$ according to $Q$ to classify $\mathbf{x}$. The true risk $R(G_Q)$ and the empirical risk $R_S(G_Q)$ of the Gibbs classifier are thus given by

$$R(G_Q) = \operatorname*{\mathbf{E}}_{h \sim Q} R(h) \quad ; \quad R_S(G_Q) = \operatorname*{\mathbf{E}}_{h \sim Q} R_S(h) \,.$$

Any bound for $R(G_Q)$ can straightforwardly be turned into a bound for the risk of the majority vote $R(B_Q)$. Indeed, whenever $B_Q$ misclassifies $\mathbf{x}$, at least half of the classifiers (under measure $Q$) misclassifies $\mathbf{x}$. It follows that the error rate of $G_Q$ is at least half of the error rate of $B_Q$. Hence $R(B_Q) \leq 2R(G_Q)$. As shown in [5], this factor of 2 can sometimes be reduced to $(1 + \epsilon)$.

## 3 PAC-Bayes Bounds and General Loss Functions

In this paper, we use the following PAC-Bayes bound which is obtained directly from Theorem 1.2.1 of [1] and Corollary 2.2 of [3] by using $1 - \exp(-x) \leq x \; \forall x \in \mathbb{R}$.

**Theorem 3.1.** *For any distribution $D$, any set $\mathcal{H}$ of classifiers, any distribution $P$ of support $\mathcal{H}$, any $\delta \in (0, 1]$, and any positive real number $C'$, we have*

$$\Pr_{S \sim D^m} \left( \forall Q \text{ on } \mathcal{H} \colon R(G_Q) \leq \frac{1}{1 - e^{-C'}} \left[ C' \cdot R_S(G_Q) + \frac{1}{m} \left[ \mathrm{KL}(Q\|P) + \ln \frac{1}{\delta} \right] \right] \right) \geq 1 - \delta,$$

*where* $\mathrm{KL}(Q\|P) \stackrel{\text{def}}{=} \operatorname*{\mathbf{E}}_{h \sim Q} \ln \frac{Q(h)}{P(h)}$ *is the Kullback-Leibler divergence between $Q$ and $P$.*

Note that the dependence on $Q$ of the upper bound on $R(G_Q)$ is realized via Gibbs' empirical risk $R_S(G_Q)$ and the PAC-Bayes regularizer $\mathrm{KL}(Q\|P)$. As in boosting, we focus on the case where the a priori defined class $\mathcal{H}$ consists (mostly) of "weak" classifiers having large risk $R(h)$. In this case, $R(G_Q)$ is (almost) always large (near 1/2) for any $Q$ *even if the majority vote $B_Q$ has null risk*. In this case the disparity between $R(B_Q)$ and $R(G_Q)$ is enormous and the upper-bound on $R(G_Q)$ has very little relevance with $R(B_Q)$. On way to obtain a more relevant bound on $R(B_Q)$ from PAC-Bayes theory is to use a loss function $\zeta_Q(\mathbf{x}, y)$ for stochastic classifiers which is distinct from the loss used for the deterministic classifiers (the zero-one loss in our case). In order to obtain a tractable optimization problem for a learning algorithm to solve, we propose here to use a loss $\zeta_Q(\mathbf{x}, y)$ which is convex in $Q$ and that upper-bounds as closely as possible the zero-one loss of the deterministic majority vote $B_Q$.

Consider $W_Q(\mathbf{x}, y) \overset{\text{def}}{=} \mathbf{E}_{h \sim Q} I(h(\mathbf{x}) \neq y)$, the $Q$-fraction of binary classifiers that err on example $(\mathbf{x}, y)$. Then, $R(G_Q) = \mathbf{E}_{(\mathbf{x}, y) \sim D} W_Q(\mathbf{x}, y)$. Following [2], we consider any non-negative convex loss $\zeta_Q(\mathbf{x}, y)$ that can be expanded in a Taylor series around $W_Q(\mathbf{x}, y) = 1/2$:

$$\zeta_Q(\mathbf{x}, y) \overset{\text{def}}{=} 1 + \sum_{k=1}^{\infty} a_k \left(2 W_Q(\mathbf{x}, y) - 1\right)^k = 1 + \sum_{k=1}^{\infty} a_k \left(\underset{h \sim Q}{\mathbf{E}} - y h(\mathbf{x})\right)^k,$$

that upper bounds the risk of the majority vote $B_Q$, *i.e.*,

$$\zeta_Q(\mathbf{x}, y) \geq I\left(W_Q(\mathbf{x}, y) > \frac{1}{2}\right) \quad \forall Q, \mathbf{x}, y.$$

It has been shown [2] that $\zeta_Q(\mathbf{x}, y)$ can be expressed in terms of the risk on example $(\mathbf{x}, y)$ of a Gibbs classifier described by a transformed posterior $\overline{Q}$ on $\mathbb{N} \times \mathcal{H}^{\infty}$, *i.e.*,

$$\zeta_Q(\mathbf{x}, y) = 1 + c_a \left[2 W_{\overline{Q}}(\mathbf{x}, y) - 1\right],$$

where $c_a \overset{\text{def}}{=} \sum_{k=1}^{\infty} |a_k|$ and where

$$W_{\overline{Q}}(\mathbf{x}, y) \overset{\text{def}}{=} \frac{1}{c_a} \sum_{k=1}^{\infty} |a_k| \underset{h_1 \sim Q}{\mathbf{E}} \cdots \underset{h_k \sim Q}{\mathbf{E}} I\left((-y)^k h_1(\mathbf{x}) \ldots h_k(\mathbf{x}) = -\operatorname{sgn}(a_k)\right).$$

Since $W_{\overline{Q}}(\mathbf{x}, y)$ is the expectation of boolean random variable, Theorem 3.1 holds if we replace $(P, Q)$ by $(\overline{P}, \overline{Q})$ with $R(G_{\overline{Q}}) \overset{\text{def}}{=} \underset{(\mathbf{x}, y) \sim D}{\mathbf{E}} W_{\overline{Q}}(\mathbf{x}, y)$ and $R_S(G_{\overline{Q}}) \overset{\text{def}}{=} \frac{1}{m} \sum_{i=1}^{m} W_{\overline{Q}}(\mathbf{x}_i, y_i)$. Moreover, it has been shown [2] that

$$\text{KL}(\overline{Q} \| \overline{P}) = \overline{k} \cdot \text{KL}(Q \| P), \quad \text{where } \overline{k} \overset{\text{def}}{=} \frac{1}{c_a} \sum_{k=1}^{\infty} |a_k| \cdot k.$$

If we define

$$\zeta_Q \overset{\text{def}}{=} \underset{(\mathbf{x}, y) \sim D}{\mathbf{E}} \zeta(\mathbf{x}, y) = 1 + c_a [2 R(G_{\overline{Q}}) - 1]$$

$$\widehat{\zeta_Q} \overset{\text{def}}{=} \frac{1}{m} \sum_{i=1}^{m} \zeta(\mathbf{x}_i, y_i) = 1 + c_a [2 R_S(G_{\overline{Q}}) - 1],$$

Theorem 3.1 gives an upper bound on $\zeta_Q$ and, consequently, on the true risk $R(B_Q)$ of the majority vote. More precisely, we have the following theorem.

**Theorem 3.2.** *For any $D$, any $\mathcal{H}$, any $P$ of support $\mathcal{H}$, any $\delta \in (0, 1]$, any positive real number $C'$, any loss function $\zeta_Q(\mathbf{x}, y)$ defined above, we have*

$$\underset{S \sim D^m}{\Pr} \left(\forall Q \text{ on } \mathcal{H} : \zeta_Q \leq g(c_a, C') + \frac{C'}{1 - e^{-C'}} \left[\widehat{\zeta_Q} + \frac{2 c_a}{m C'} \left[\overline{k} \cdot \text{KL}(Q \| P) + \ln \frac{1}{\delta}\right]\right]\right) \geq 1 - \delta,$$

*where $g(c_a, C') \overset{\text{def}}{=} 1 - c_a + \frac{C'}{1 - e^{-C'}} \cdot (c_a - 1)$.*

## 4  Bound Minimization Learning Algorithms

The task of the learner is to find the posterior $Q$ that minimizes the upper bound on $\zeta_Q$ for a fixed loss function given by the coefficients $\{a_k\}_{k=1}^{\infty}$ of the Taylor series expansion for $\zeta_Q(\mathbf{x}, y)$. Finding $Q$ that minimizes the upper bound given by Theorem 3.2 is equivalent to finding $Q$ that minimizes

$$f(Q) \overset{\text{def}}{=} C \sum_{i=1}^{m} \zeta_Q(\mathbf{x}_i, y_i) + \text{KL}(Q \| P),$$

where $C \overset{\text{def}}{=} C'/(2 c_a \overline{k})$.

To compare the proposed learning algorithms with AdaBoost, we will consider, for $\zeta_Q(\mathbf{x}, y)$, the *exponential loss* given by

$$\exp\left(-\frac{1}{\gamma} y \sum_{h \in \mathcal{H}} Q(h) h(\mathbf{x})\right) \;=\; \exp\left(\frac{1}{\gamma}\left[2W_Q(\mathbf{x}, y) - 1\right]\right).$$

For this choice of loss, we have $c_a = e^{\gamma^{-1}} - 1$ and $\overline{k} = \gamma^{-1}/(1 - e^{-\gamma^{-1}})$. Because of its simplicity, we will also consider, for $\zeta_Q(\mathbf{x}, y)$, the *quadratic loss* given by

$$\left(\frac{1}{\gamma} y \sum_{h \in \mathcal{H}} Q(h) h(\mathbf{x}) - 1\right)^2 \;=\; \left(\frac{1}{\gamma}\left[1 - 2W_Q(\mathbf{x}, y)\right] - 1\right)^2.$$

For this choice of loss, we have $c_a = 2\gamma^{-1} + \gamma^{-2}$ and $\overline{k} = (2\gamma + 2)/(2\gamma + 1)$. Note that this loss has the minimum value of zero for examples having a margin $y \sum_{h \in \mathcal{H}} Q(h) h(\mathbf{x}) = \gamma$.

With these two choices of loss functions, $\zeta_Q(\mathbf{x}, y)$ is convex in $Q$. Moreover, $\mathrm{KL}(Q\|P)$ is also convex in $Q$. Since a sum of convex functions is also convex, it follows that objective function $f$ is convex in $Q$ (which has a convex domain). Consequently, $f$ has a single local minimum which coincides with the global minimum. We therefore propose to minimize $f$ coordinate-wise, similarly as it is done for AdaBoost [7]. However, to ensure that $Q$ is a distribution (*i.e.*, that $\sum_{h \in \mathcal{H}} Q(h) = 1$), each coordinate minimization will consist of a transfer of weight from one classifier to another.

## 4.1 Quasi Uniform Posteriors

We consider learning algorithms that work in a space $\mathcal{H}$ of binary classifiers such that for each $h \in \mathcal{H}$, the boolean complement of $h$ is also in $\mathcal{H}$. More specifically, we have $\mathcal{H} = \{h_1, \ldots, h_n, h_{n+1}, \ldots, h_{2n}\}$ where $h_i(\mathbf{x}) = -h_{n+i}(\mathbf{x}) \; \forall \mathbf{x} \in \mathcal{X}$ and $\forall i \in \{1, \ldots, n\}$. We thus say that $(h_i, h_{n+i})$ constitutes a *boolean complement pair* of classifiers.

We consider a uniform prior distribution $P$ over $\mathcal{H}$, *i.e.*, $P_i = \frac{1}{2n} \; \forall i \in \{1, \ldots, 2n\}$.

The posterior distribution $Q$ over $\mathcal{H}$ is constrained to be *quasi uniform*. By this, we mean that $Q_i + Q_{i+n} = \frac{1}{n} \; \forall i \in \{1, \ldots, n\}$, *i.e.*, the total weight assigned to each boolean complement pair of classifiers is fixed to $1/n$. Let $w_i \stackrel{\text{def}}{=} Q_i - Q_{i+n} \; \forall i \in \{1, \ldots, n\}$. Then $w_i \in [-1/n, +1/n] \; \forall i \in \{1, \ldots, n\}$ whereas $Q_i \in [0, 1/n] \; \forall i \in \{1, \ldots, 2n\}$.

For any quasi uniform $Q$, the output $B_Q(\mathbf{x})$ of the majority vote on any example $\mathbf{x}$ is given by

$$B_Q(\mathbf{x}) \;=\; \mathrm{sgn}\left(\sum_{i=1}^{2n} Q_i h_i(\mathbf{x})\right) \;=\; \mathrm{sgn}\left(\sum_{i=1}^{n} w_i h_i(\mathbf{x})\right) \stackrel{\text{def}}{=} \mathrm{sgn}\left(\mathbf{w} \cdot \mathbf{h}(\mathbf{x})\right).$$

Consequently, the set of majority votes $B_Q$ over quasi uniform posteriors is isomorphic to the set of linear separators with real weights. *There is thus no loss of discriminative power if we restrict ourselves to quasi uniform posteriors.*

Since all loss functions that we consider are functions of $2W_Q(\mathbf{x}, y) - 1 = -y \sum_i Q_i h_i(\mathbf{x})$, they are thus functions of $y\mathbf{w} \cdot \mathbf{h}(\mathbf{x})$. Hence we will often write $\zeta(y\mathbf{w} \cdot \mathbf{h}(\mathbf{x}))$ for $\zeta_Q(\mathbf{x}, y)$.

The basic iteration for the learning algorithm consists of choosing (at random) a boolean complement pair of classifiers, call it $(h_1, h_{n+1})$, and then attempting to change only $Q_1, Q_{n+1}, w_1$ according to:

$$Q_1 \leftarrow Q_1 + \frac{\delta}{2} \quad ; \quad Q_{n+1} \leftarrow Q_{n+1} - \frac{\delta}{2} \quad ; \quad w_1 \leftarrow w_1 + \delta \,, \tag{1}$$

for some optimally chosen value of $\delta$.

Let $Q_\delta$ and $\mathbf{w}_\delta$ be, respectively, the new posterior and the new weight vector obtained with such a change. The above-mentioned convex properties of objective function $f$ imply that we only need to look for the value of $\delta^*$ satisfying

$$\frac{df(Q_\delta)}{d\delta} = 0 \,. \tag{2}$$

If $w_1 + \delta^* > 1/n$, then $w_1 \leftarrow 1/n$, $Q_1 \leftarrow 1/n$, $Q_{n+1} \leftarrow 0$. If $w_1 + \delta^* < -1/n$, then $w_1 \leftarrow -1/n$, $Q_1 \leftarrow 0$, $Q_{n+1} \leftarrow 1/n$. Otherwise, we accept the change described by Equation 1 with $\delta = \delta^*$.

For objective function $f$ we simply have

$$\frac{df(Q_\delta)}{d\delta} = Cm\frac{\widehat{d\zeta_{Q_\delta}}}{d\delta} + \frac{d\text{KL}(Q_\delta\|P)}{d\delta}\,, \tag{3}$$

where

$$\begin{aligned}\frac{d\text{KL}(Q_\delta\|P)}{d\delta} &= \frac{d}{d\delta}\left[\left(Q_1 + \frac{\delta}{2}\right)\ln\frac{Q_1 + \frac{\delta}{2}}{\frac{1}{2n}} + \left(Q_{n+1} - \frac{\delta}{2}\right)\ln\frac{Q_{n+1} - \frac{\delta}{2}}{\frac{1}{2n}}\right]\\ &= \frac{1}{2}\ln\left[\frac{Q_1 + \delta/2}{Q_{n+1} - \delta/2}\right]\,. \end{aligned} \tag{4}$$

For the quadratic loss, we find

$$m\frac{\widehat{d\zeta_{Q_\delta}}}{d\delta} = \frac{2m\delta}{\gamma^2} + \frac{2}{\gamma^2}\sum_{i=1}^{m} D_{\mathbf{w}}^{ql}(i)y_i h_1(\mathbf{x}_i)\,, \tag{5}$$

where

$$D_{\mathbf{w}}^{ql}(i) \stackrel{\text{def}}{=} y_i\mathbf{w}\cdot\mathbf{h}(\mathbf{x}_i) - \gamma\,. \tag{6}$$

Consequently, for the quadratic loss case, the optimal value $\delta^*$ satisfies

$$\frac{2Cm\delta}{\gamma^2} + \frac{2C}{\gamma^2}\sum_{i=1}^{m} D_{\mathbf{w}}^{ql}(i)y_i h_1(\mathbf{x}_i) + \frac{1}{2}\ln\left[\frac{Q_1 + \delta/2}{Q_{n+1} - \delta/2}\right] = 0\,. \tag{7}$$

For the exponential loss, we find

$$m\frac{\widehat{d\zeta_{Q_\delta}}}{d\delta} = \frac{e^{\delta/\gamma}}{\gamma}\sum_{i=1}^{m} D_{\mathbf{w}}^{el}(i)I(h_1(\mathbf{x}_i) \neq y_i) - \frac{e^{-\delta/\gamma}}{\gamma}\sum_{i=1}^{m} D_{\mathbf{w}}^{el}(i)I(h_1(\mathbf{x}_i) = y_i)\,, \tag{8}$$

where

$$D_{\mathbf{w}}^{el}(i) \stackrel{\text{def}}{=} \exp\left(-\frac{1}{\gamma}y_i\mathbf{w}\cdot\mathbf{h}(\mathbf{x}_i)\right)\,. \tag{9}$$

Consequently, for the exponential loss case, the optimal value $\delta^*$ satisfies

$$\begin{aligned}\frac{Ce^{\delta/\gamma}}{\gamma}\sum_{i=1}^{m} &D_{\mathbf{w}}^{el}(i)I(h_1(\mathbf{x}_i) \neq y_i)\\ &- \frac{Ce^{-\delta/\gamma}}{\gamma}\sum_{i=1}^{m} D_{\mathbf{w}}^{el}(i)I(h_1(\mathbf{x}_i) = y_i) + \frac{1}{2}\ln\left[\frac{Q_1 + \delta/2}{Q_{n+1} - \delta/2}\right] = 0\,. \end{aligned} \tag{10}$$

After changing $w_1$, we need to recompute[1] $D_{\mathbf{w}}(i)\ \forall i \in \{1,\ldots,m\}$. This can be done with the following update rules.

$$D_{\mathbf{w}}^{ql}(i) \leftarrow D_{\mathbf{w}}^{ql}(i) + y_i h_1(\mathbf{x}_i)\delta \quad \text{(quadratic loss case)} \tag{11}$$

$$D_{\mathbf{w}}^{el}(i) \leftarrow D_{\mathbf{w}}^{el}(i)e^{-\frac{1}{\gamma}y_i h_1(\mathbf{x}_i)\delta} \quad \text{(exponential loss case)}\,. \tag{12}$$

Since, initially we have

$$D_{\mathbf{w}}^{ql}(i) = -\gamma\ \forall i \in \{1,\ldots,m\} \quad \text{(quadratic loss case)} \tag{13}$$

$$D_{\mathbf{w}}^{el}(i) = 1\ \forall i \in \{1,\ldots,m\} \quad \text{(exponential loss case)}\,, \tag{14}$$

the dot product present in Equations 6 and 9 never needs to be computed. Consequently, updating $D_{\mathbf{w}}$ takes $\Theta(m)$ time.

The computation of the summations over the $m$ examples in Equation 7 or 10 takes $\Theta(m)$ time. Once these summations are computed, solving Equation 7 or 10 takes $\Theta(1)$ time. Consequently, it takes $\Theta(m)$ time to perform one basic iteration of the learning algorithm which consist of (1) solving Equation 7 or 10 to find $\delta^*$, (2) modifying $w_1, Q_1, Q_{n+1}$, and (3) updating $D_{\mathbf{w}}$ according to Equation 11 or 12. The complete algorithm, called *f minimization*, is described by the pseudo code of Algorithm 1.

**Algorithm 1** : $f$ minimization

---

1: **Initialization:** Let $Q_i = Q_{n+i} = \frac{1}{2n}$, $w_i = 0$, $\forall i \in \{1, \ldots, n\}$.

   Initialize $D_{\mathbf{w}}$ according to Equation 13 or 14.

2: **repeat**

3:    Choose at random $h \in \mathcal{H}$ and call it $h_1$ ($h_{n+1}$ is then the boolean complement of $h_1$).

4:    Find $\delta^*$ that solves Equation 7 or 10.

5:    If $[\frac{-1}{n} < w_1 + \delta^* < \frac{1}{n}]$ then $Q_1 \leftarrow Q_1 + \delta/2$; $Q_{n+1} \leftarrow Q_{n+1} - \delta/2$; $w_1 \leftarrow w_1 + \delta$.

6:    If $[w_1 + \delta^* \geq \frac{1}{n}]$ then $Q_1 \leftarrow \frac{1}{n}$; $Q_{n+1} \leftarrow 0$; $w_1 \leftarrow \frac{1}{n}$.

7:    If $[w_1 + \delta^* \leq \frac{-1}{n}]$ then $Q_1 \leftarrow 0$; $Q_{n+1} \leftarrow \frac{1}{n}$; $w_1 \leftarrow \frac{-1}{n}$.

8:    Update $D_{\mathbf{w}}$ according to Equation 11 or 12.

9: **until** Convergence

---

The repeat-until loop in Algorithm 1 was implemented as follows. We first mix at random the $n$ boolean complement pairs of classifiers and then go sequentially over each pair $(h_i, h_{n+i})$ to update $w_i$ and $D_{\mathbf{w}}$. We repeat this sequence until no weight change exceeds a specified small number $\epsilon$.

## 4.2 From $\mathrm{KL}(Q\|P)$ to $\ell_p$ Regularization

We can recover $\ell_2$ regularization if we upper-bound $\mathrm{KL}(Q\|P)$ by a quadratic function. Indeed, if we use

$$q \ln q + \left(\frac{1}{n} - q\right) \ln \left(\frac{1}{n} - q\right) \ \leq \ \frac{1}{n} \ln \frac{1}{2n} + 4n \left(q - \frac{1}{2n}\right)^2 \ \forall q \in [0, 1/n], \qquad (15)$$

we obtain, for the uniform prior $P_i = 1/(2n)$,

$$\begin{aligned} \mathrm{KL}(Q\|P) &= \ln(2n) + \sum_{i=1}^{n} \left[ Q_i \ln Q_i + \left(\frac{1}{n} - Q_i\right) \ln \left(\frac{1}{n} - Q_i\right) \right] \\ &\leq 4n \sum_{i=1}^{n} \left(Q_i - \frac{1}{2n}\right)^2 \ = \ n \sum_{i=1}^{n} w_i^2 . \end{aligned} \qquad (16)$$

With this approximation, the objective function to minimize becomes

$$f_{\ell_2}(\mathbf{w}) \ = \ C'' \sum_{i=1}^{m} \zeta \left(\frac{1}{\gamma} y_i \mathbf{w} \cdot \mathbf{h}(\mathbf{x}_i)\right) + \|\mathbf{w}\|_2^2 , \qquad (17)$$

subject to the $\ell_\infty$ constraint $|w_j| \leq 1/n \ \forall j \in \{1, \ldots, n\}$. Here $\|\mathbf{w}\|_2$ denotes the Euclidean norm of $\mathbf{w}$ and $\zeta(x) = (x-1)^2$ for the quadratic loss and $e^{-x}$ for the exponential loss.

If, instead, we minimize $f_{\ell_2}$ for $\mathbf{v} \stackrel{\text{def}}{=} \mathbf{w}/\gamma$ and remove the $\ell_\infty$ constraint, we recover *exactly* ridge regression for the quadratic loss case and $\ell_2$-regularized boosting for the exponential loss case.

We can obtain a $\ell_1$-regularized version of Equation 17 by repeating the above steps and using $4n \left(q - \frac{1}{2n}\right)^2 \ \leq \ 2 \left|q - \frac{1}{2n}\right| \ \forall q \in [0, 1/n]$ since, in that case, we find that $\mathrm{KL}(Q\|P) \leq \sum_{i=1}^{n} |w_i| \stackrel{\text{def}}{=} \|\mathbf{w}\|_1$.

To sum up, the KL-regularized objective function $f$ immediately follows from PAC-Bayes theory and $\ell_p$ regularization is obtained from a *relaxation* of $f$. Consequently, PAC-Bayes theory favors the use of KL regularization if the goal of the learner is to produce a weighted majority vote with good generalization.[2]

# 5 Empirical Results

For the sake of comparison, all learning algorithms of this subsection are producing a weighted majority vote classifier on the set of basis functions $\{h_1, \ldots, h_n\}$ known as *decision stumps*. Each decision stump $h_i$ is a threshold classifier that depends on a single attribute: its output is $+b$ if the tested attribute exceeds a threshold value $t$, and $-b$ otherwise, where $b \in \{-1, +1\}$. For each attribute, at most ten equally-spaced possible values for $t$ were determined *a priori*. Recall that, although Algorithm 1 needs a set $\mathcal{H}$ of $2n$ classifiers containing $n$ boolean complement pairs, it outputs a majority vote with $n$ real-valued weights defined on $\{h_1, \ldots, h_n\}$.

The results obtained for all tested algorithms are summarized in Table 1. We have compared Algorithm 1 with quadratic loss (KL-QL) and exponential loss (KL-EL) to AdaBoost [7] (AdB) and ridge regression (RR).

Except for MNIST, all data sets were taken from the UCI repository. Each data set was randomly split into a training set $S$ of $|S|$ examples and a testing set $T$ of $|T|$ examples. The number $a$ of attributes for each data set is also specified in Table 1. For AdaBoost, the number of boosting rounds was fixed to 200. For all algorithms, $R_T$ refers to the frequency of errors, measured on the testing set $T$.

In addition to this, the "$C$ and "$\gamma$" columns in Table 1 refer, respectively, to the $C$ value of the objective function $f$ and to the $\gamma$ parameter present in the loss functions. These hyperparameters were determined from the training set only by performing the 10-fold cross validation (CV) method. The hyperparameters that gave the smallest 10-fold CV error were then used to train the Algorithms on the whole training set and the resulting classifiers were then run on the testing set.

Table 1: Summary of results.

| Dataset | | | | (1) AdB | (2) RR | | (3) KL-EL | | | (4) KL-QL | | | SSB |
|---|---|---|---|---|---|---|---|---|---|---|---|---|---|
| Name | $|S|$ | $|T|$ | $a$ | $R_T$ | $R_T$ | $C$ | $R_T$ | $C$ | $\gamma$ | $R_T$ | $C$ | $\gamma$ | |
| BreastCancer | 343 | 340 | 9 | 0.053 | 0.050 | 10 | **0.047** | 0.1 | 0.1 | **0.047** | 0.02 | 0.4 | |
| Liver | 170 | 175 | 6 | 0.320 | 0.309 | 5 | 0.360 | 0.5 | 0.02 | **0.286** | 0.02 | 0.3 | |
| Credit-A | 353 | 300 | 15 | 0.170 | **0.157** | 2 | 0.227 | 0.1 | 0.2 | 0.183 | 0.02 | 0.05 | |
| Glass | 107 | 107 | 9 | **0.178** | 0.206 | 5 | 0.187 | 500 | 0.01 | 0.196 | 0.02 | 0.01 | |
| Haberman | 144 | 150 | 3 | 0.260 | 0.273 | 100 | **0.253** | 500 | 0.2 | 0.260 | 0.02 | 0.5 | |
| Heart | 150 | 147 | 13 | 0.252 | 0.197 | 1 | 0.211 | 0.2 | 0.1 | **0.177** | 0.05 | 0.2 | |
| Ionosphere | 176 | 175 | 34 | 0.120 | 0.131 | 0.05 | 0.120 | 20 | 0.0001 | **0.097** | 0.2 | 0.1 | |
| Letter:AB | 500 | 1055 | 16 | 0.010 | **0.004** | 0.5 | 0.006 | 0.1 | 0.02 | 0.006 | 1000 | 0.1 | |
| Letter:DO | 500 | 1058 | 16 | 0.036 | 0.026 | 0.05 | **0.019** | 500 | 0.01 | 0.020 | 0.02 | 0.05 | |
| Letter:OQ | 500 | 1036 | 16 | **0.038** | 0.045 | 0.5 | 0.043 | 10 | 0.0001 | 0.047 | 0.1 | 0.05 | |
| MNIST:0vs8 | 500 | 1916 | 784 | 0.008 | 0.015 | 0.05 | **0.006** | 500 | 0.001 | 0.015 | 0.2 | 0.02 | $(3) < (2, 4)$ |
| MNIST:1vs7 | 500 | 1922 | 784 | 0.013 | **0.012** | 1 | 0.014 | 500 | 0.02 | 0.014 | 1000 | 0.1 | |
| MNIST:1vs8 | 500 | 1936 | 784 | 0.025 | 0.024 | 0.2 | **0.016** | 0.2 | 0.001 | 0.031 | 1 | 0.02 | $(3) < (4)$ |
| MNIST:2vs3 | 500 | 1905 | 784 | 0.047 | 0.033 | 0.2 | 0.035 | 500 | 0.0001 | **0.029** | 0.02 | 0.05 | $(4) < (1)$ |
| Mushroom | 4062 | 4062 | 22 | **0.000** | 0.001 | 0.5 | **0.000** | 10 | 0.001 | **0.000** | 1000 | 0.02 | |
| Ringnorm | 3700 | 3700 | 20 | 0.043 | 0.037 | 0.05 | **0.025** | 500 | 0.01 | 0.039 | 0.05 | 0.05 | $(3) < (1, 2, 4)$ |
| Sonar | 104 | 104 | 60 | 0.231 | 0.192 | 0.05 | 0.135 | 500 | 0.05 | **0.115** | 1000 | 0.1 | |
| Usvotes | 235 | 200 | 16 | **0.055** | 0.060 | 2 | 0.060 | 0.5 | 0.1 | **0.055** | 1000 | 0.05 | |
| Waveform | 4000 | 4000 | 21 | 0.085 | **0.079** | 0.02 | 0.080 | 0.2 | 0.05 | 0.080 | 0.02 | 0.05 | |
| Wdbc | 285 | 284 | 30 | 0.049 | 0.049 | 0.2 | **0.039** | 500 | 0.02 | 0.046 | 1000 | 0.1 | |

We clearly see that the cross-validation method generally chooses very small values for $\gamma$. This, in turn, gives a risk bound (computed from Theorem 3.2) having very large values (results not shown here). We have also tried to choose $C$ and $\gamma$ from the risk bound values.[3] This method for selecting hyperparameters turned out to produce classifiers having larger testing errors (results not shown here).

To determine whether or not a difference of empirical risk measured on the testing set $T$ is statistically significant, we have used the test set bound method of [4] (based on the binomial tail inversion)

with a confidence level of $95\%$. It turns out that no algorithm has succeeded in choosing a majority vote classifier which was statistically significantly better (SSB) than the one chosen by another algorithm *except* for the 4 cases that are listed in the column "SSB" of Table 1. We see that on these cases, Algorithm 1 turned out to be statistically significantly better.

## 6   Conclusion

Our numerical results indicate that Algorithm 1 generally outperforms AdaBoost and ridge regression when the hyperparameters $C$ and $\gamma$ are chosen by cross-validation. This indicates that the empirical loss $\widehat{\zeta_Q}$ and the $\mathrm{KL}(Q\|P)$ regularizer that are present in the PAC-Bayes bound of Theorem 3.2 are key ingredients for learning algorithms to focus on. The fact that cross-validation turns out to be more efficient than Theorem 3.2 at selecting good values for hyperparameters indicates that PAC-Bayes theory does not yet capture quantitatively the proper tradeoff between $\widehat{\zeta_Q}$ and $\mathrm{KL}(Q\|P)$ that learners should optimize on the trading data. However, we feel that it is important to pursue this research direction since it could potentially eliminate the need to perform the time-consuming cross-validation method for selecting hyperparameters and provide better guarantees on the generalization error of classifiers output by learning algorithms. In short, it could perhaps yield the best generic optimization problem for learning.

**Acknowledgments**

Work supported by NSERC discovery grants 122405 (M.M.) and 262067 (F.L.).

## Footnotes

[1] $D_{\mathbf{w}}(i)$ stands for either $D_{\mathbf{w}}^{ql}(i)$ or $D_{\mathbf{w}}^{el}(i)$.

[2]Interestingly, [9] has recently proposed a KL-regularized version of LPBoost but their objective function was not derived from a uniform risk bound.

[3]From the standard union bound argument, the bound of Theorem 3.2 holds simultaneously for $k$ different choices of $(\gamma, C)$ if we replace $\delta$ by $\delta/k$.

## References

[1] Olivier Catoni. *PAC-Bayesian surpevised classification: the thermodynamics of statistical learning*. Monograph series of the Institute of Mathematical Statistics, http://arxiv.org/abs/0712.0248, December 2007.

[2] Pascal Germain, Alexandre Lacasse, François Laviolette, and Mario Marchand. A pac-bayes risk bound for general loss functions. In B. Schölkopf, J. Platt, and T. Hoffman, editors, *Advances in Neural Information Processing Systems 19*, pages 449–456. MIT Press, Cambridge, MA, 2007.

[3] Pascal Germain, Alexandre Lacasse, François Laviolette, and Mario Marchand. PAC-Bayesian learning of linear classifiers. In Léon Bottou and Michael Littman, editors, *Proceedings of the 26th International Conference on Machine Learning*, pages 353–360, Montreal, June 2009. Omnipress.

[4] John Langford. Tutorial on practical prediction theory for classification. *Journal of Machine Learning Research*, 6:273–306, 2005.

[5] John Langford and John Shawe-Taylor. PAC-Bayes & margins. In S. Thrun S. Becker and K. Obermayer, editors, *Advances in Neural Information Processing Systems 15*, pages 423–430. MIT Press, Cambridge, MA, 2003.

[6] David McAllester. PAC-Bayesian stochastic model selection. *Machine Learning*, 51:5–21, 2003.

[7] Robert E. Schapire, Yoav Freund, Peter Bartlett, and Wee Sun Lee. Boosting the margin: A new explanation for the effectiveness of voting methods. *The Annals of Statistics*, 26:1651–1686, 1998.

[8] Matthias Seeger. PAC-Bayesian generalization bounds for gaussian processes. *Journal of Machine Learning Research*, 3:233–269, 2002.

[9] Manfred K. Warmuth, Karen A. Glocer, and S.V.N. Vishwanathan. Entropy regularized LP-Boost. In *Proceedings of the 2008 conference on Algorithmic Learning Theory, Springer LNAI 5254,*, pages 256–271, 2008.

